# Adaptive Nearest Neighbor Classification using Support Vector Machines

**Carlotta Domeniconi, Dimitrios Gunopulos**
Dept. of Computer Science, University of California, Riverside, CA 92521
{*carlotta,dg*}*@cs.ucr.edu*

## Abstract

The nearest neighbor technique is a simple and appealing method to address classification problems. It relies on the assumption of locally constant class conditional probabilities. This assumption becomes invalid in high dimensions with a finite number of examples due to the curse of dimensionality. We propose a technique that computes a locally flexible metric by means of Support Vector Machines (SVMs). The maximum margin boundary found by the SVM is used to determine the most discriminant direction over the query's neighborhood. Such direction provides a local weighting scheme for input features. We present experimental evidence of classification performance improvement over the SVM algorithm alone and over a variety of adaptive learning schemes, by using both simulated and real data sets.

## 1   Introduction

In a classification problem, we are given $J$ classes and $l$ training observations. The training observations consist of $n$ feature measurements $\mathbf{x} = (x_1, \cdots, x_n)^T \in \Re^n$ and the known class labels $j = 1, \ldots, J$. The goal is to predict the class label of a given query $\mathbf{q}$.

The $K$ nearest neighbor classification method [4, 13, 16] is a simple and appealing approach to this problem: it finds the $K$ nearest neighbors of $\mathbf{q}$ in the training set, and then predicts the class label of $\mathbf{q}$ as the most frequent one occurring in the $K$ neighbors. It has been shown [5, 8] that the one nearest neighbor rule has asymptotic error rate that is at most twice the Bayes error rate, independent of the distance metric used. The nearest neighbor rule becomes less appealing with finite training samples, however. This is due to the curse of dimensionality [2]. Severe bias can be introduced in the nearest neighbor rule in a high dimensional input feature space with finite samples. As such, the choice of a distance measure becomes crucial in determining the outcome of nearest neighbor classification. The commonly used Euclidean distance implies that the input space is isotropic, which is often invalid and generally undesirable in many practical applications.

Several techniques [9, 10, 7] have been proposed to try to minimize bias in high dimensions by using locally adaptive mechanisms. The "lazy learning" approach used

by these methods, while appealing in many ways, requires a considerable amount of on-line computation, which makes it difficult for such techniques to scale up to large data sets. The feature weighting scheme they introduce, in fact, is query based and is applied on-line when the test point is presented to the "lazy learner". In this paper we propose a locally adaptive metric classification method which, although still founded on a query based weighting mechanism, computes off-line the information relevant to define local weights.

Our technique uses support vector machines (SVMs) as a guidance for the process of defining a local flexible metric. SVMs have been successfully used as a classification tool in a variety of areas [11, 3, 14], and the maximum margin boundary they provide has been proved to be optimal in a structural risk minimization sense. The solid theoretical foundations that have inspired SVMs convey desirable computational and learning theoretic properties to the SVM's learning algorithm, and therefore SVMs are a natural choice for seeking local discriminant directions between classes.

The solution provided by SVMs allows to determine locations in input space where class conditional probabilities are likely to be not constant, and guides the extraction of local information in such areas. This process produces highly stretched neighborhoods along boundary directions when the query is close to the boundary. As a result, the class conditional probabilities tend to be constant in the modified neighborhoods, whereby better classification performance can be achieved. The amount of elongation-constriction decays as the query moves further from the boundary vicinity.

## 2  Feature Weighting

SVMs classify patterns according to the $sign(f(\mathbf{x}))$, where $f(\mathbf{x}) = \sum_{i=1}^{l} \alpha_i y_i K(\mathbf{x}_i, \mathbf{x}) - b$, $K(\mathbf{x}, \mathbf{y}) = \phi^T(\mathbf{x}) \cdot \phi(\mathbf{y})$ (*kernel function*), and $\phi : \Re^n \to \Re^N$ is a mapping of the input vectors into a higher dimensional feature space. Here we assume $\mathbf{x}_i \in \Re^n$, $i = 1, \ldots, l$, and $y_i \in \{-1, 1\}$. Clearly, in the general case of a non-linear feature mapping $\phi$, the SVM classifier gives a non-linear boundary $f(\mathbf{x}) = 0$ in input space. The gradient vector $\mathbf{n_d} = \nabla_{\mathbf{d}} f$, computed at any point $\mathbf{d}$ of the level curve $f(\mathbf{x}) = 0$, gives the perpendicular direction to the decision boundary in input space at $\mathbf{d}$. As such, the vector $\mathbf{n_d}$ identifies the orientation in input space on which the projected training data are well separated, locally over $\mathbf{d}$'s neighborhood. Therefore, the orientation given by $\mathbf{n_d}$, and any orientation close to it, is highly informative for the classification task at hand, and we can use such information to define a local measure of feature relevance.

Let $\mathbf{q}$ be a query point whose class label we want to predict. Suppose $\mathbf{q}$ is close to the boundary, which is where class conditional probabilities become locally non uniform, and therefore estimation of local feature relevance becomes crucial. Let $\mathbf{d}$ be the closest point to $\mathbf{q}$ on the boundary $f(\mathbf{x}) = 0$: $\mathbf{d} = \arg\min_{\mathbf{p}} \|\mathbf{q} - \mathbf{p}\|$, subject to the constraint $f(\mathbf{p}) = 0$. Then we know that the gradient $\mathbf{n_d}$ identifies a direction along which data points between classes are well separated.

As a consequence, the subspace spanned by the orientation $\mathbf{n_d}$, locally at $\mathbf{q}$, is likely to contain points having the same class label as $\mathbf{q}$. Therefore, when applying a nearest neighbor rule at $\mathbf{q}$, we desire to stay close to $\mathbf{q}$ along the $\mathbf{n_d}$ direction, because that is where it is likely to find points similar to $\mathbf{q}$ in terms of class posterior probabilities. Distances should be constricted (large weight) along $\mathbf{n_d}$ and along directions close to it. The farther we move from the $\mathbf{n_d}$ direction, the less discriminant the correspondent orientation becomes. This means that class labels are likely not to change along those orientations, and distances should be elongated

(small weight), thus including in $\mathbf{q}$'s neighborhood points which are likely to be similar to $\mathbf{q}$ in terms of the class conditional probabilities.

Formally, we can measure how close a direction $\mathbf{t}$ is to $\mathbf{n_d}$ by considering the dot product $\mathbf{n_d}^T \cdot \mathbf{t}$. In particular, by denoting with $\mathbf{u}_j$ the unit vector along input feature $j$, for $j = 1, \ldots, n$, we can define a measure of relevance for feature $j$, locally at $\mathbf{q}$ (and therefore at $\mathbf{d}$), as $R_j(\mathbf{q}) \equiv |\mathbf{u}_j^T \cdot \mathbf{n_d}| = |n_{\mathbf{d},j}|$, where $\mathbf{n_d} = (n_{\mathbf{d},1}, \ldots, n_{\mathbf{d},n})^T$.

The measure of feature relevance, as a weighting scheme, can then be given by the following exponential weighting scheme: $w_j(\mathbf{q}) = exp(AR_j(\mathbf{q}))/\sum_{i=1}^n exp(AR_i(\mathbf{q}))$, where $A$ is a parameter that can be chosen to maximize (minimize) the influence of $R_j$ on $w_j$. When $A = 0$ we have $w_j = 1/n$, thereby ignoring any difference between the $R_j$'s. On the other hand, when $A$ is large a change in $R_j$ will be exponentially reflected in $w_j$. The exponential weighting scheme conveys stability to the method by preventing neighborhoods to extend infinitely in any direction. This is achieved by avoiding zero weights, which would instead be allowed by linear or quadratic weightings. Thus, the exponential weighting scheme can be used as weights associated with features for weighted distance computation $D(\mathbf{x}, \mathbf{y}) = \sqrt{\sum_{i=1}^n w_i(x_i - y_i)^2}$. These weights enable the neighborhood to elongate less important feature dimensions, and, at the same time, to constrict the most influential ones. Note that the technique is *query-based* because weightings depend on the query.

## 3 Local Flexible Metric Classification based on SVMs

To estimate the orientation of local boundaries, we move from the query point along the input axes at distances proportional to a given small step (whose initial value can be arbitrarily small, and doubled at each iteration till the boundary is crossed). We stop as soon as the boundary is crossed along an input axis $i$, i.e. when a point $\mathbf{p}_i$ is reached that satisfies the condition $sign(f(\mathbf{q})) \times sign(f(\mathbf{p}_i)) = -1$. Given $\mathbf{p}_i$, we can get arbitrarily close to the boundary by moving at (arbitrarily) small steps along the segment that joins $\mathbf{p}_i$ to $\mathbf{q}$.

Let us denote with $\mathbf{d}_i$ the intercepted point on the boundary along direction $i$. We then approximate $\mathbf{n_d}$ with the gradient vector $\mathbf{n}_{\mathbf{d}_i} = \nabla_{\mathbf{d}_i} f$, computed at $\mathbf{d}_i$.

We desire that the parameter $A$ in the exponential weighting scheme increases as the distance of $\mathbf{q}$ from the boundary decreases. By using the knowledge that support vectors are mostly located around the boundary surface, we can estimate how close a query point $\mathbf{q}$ is to the boundary by computing its distance from the closest non bounded support vector: $B_{\mathbf{q}} = \min_{\mathbf{s}_i} \|\mathbf{q} - \mathbf{s}_i\|$, where the minimum is taken over the non bounded ($0 < \alpha_i < C$) support vectors $\mathbf{s}_i$. Following the same principle, in [1] the spatial resolution around the boundary is increased by enlarging volume elements locally in neighborhoods of support vectors. Then, we can achieve our goal by setting $A = D - B_{\mathbf{q}}$, where $D$ is a constant input parameter of the algorithm. In our experiments we set $D$ equal to the approximated average distance between the training points $\mathbf{x}_k$ and the boundary: $D = \frac{1}{l} \sum_{\mathbf{x}_k} \{\min_{\mathbf{s}_i} \|\mathbf{x}_k - \mathbf{s}_i\|\}$. If $A$ becomes negative it is set to zero.

By doing so the value of $A$ nicely adapts to each query point according to its location with respect to the boundary. The closer $\mathbf{q}$ is to the decision boundary, the higher the effect of the $R_j$'s values will be on distances computation.

We observe that this principled guideline for setting the parameters of our technique takes advantage of the sparseness representation of the solution provided by the SVM. In fact, for each query point $\mathbf{q}$, in order to compute $B_{\mathbf{q}}$ we only need to consider the support vectors, whose number is typically small compared to the

**Input**: Decision boundary $f(\mathbf{x}) = 0$ produced by a SVM; query point $\mathbf{q}$ and parameter $K$.

1. Compute the approximated closest point $\mathbf{d}_i$ to $\mathbf{q}$ on the boundary;
2. Compute the gradient vector $\mathbf{n}_{\mathbf{d}_i} = \nabla_{\mathbf{d}_i} f$;
3. Set feature relevance values $R_j(\mathbf{q}) = |n_{\mathbf{d}_i, j}|$ for $j = 1, \ldots, n$;
4. Estimate the distance of $\mathbf{q}$ from the boundary as: $B_{\mathbf{q}} = \min_{\mathbf{s}_i} \|\mathbf{q} - \mathbf{s}_i\|$;
5. Set $A = D - B_{\mathbf{q}}$, where $D = \frac{1}{l} \sum_{\mathbf{x}_k} \{\min_{\mathbf{s}_i} \|\mathbf{x}_k - \mathbf{s}_i\|\}$;
6. Set $w_j(\mathbf{q}) = exp(AR_j(\mathbf{q}))/\sum_{i=1}^n exp(AR_i(\mathbf{q}))$, for $j = 1, \ldots, n$;
7. Use the resulting $\mathbf{w}$ for $K$-nearest neighbor classification at the query point $\mathbf{q}$.

Figure 1: The LFM-SVM algorithm

total number of training examples. Furthermore, the computation of $D$'s value is carried out once and off-line.

The resulting local flexible metric technique based on SVMs (LFM-SVM) is summarized in Figure 1. The algorithm has only one adjustable tuning parameter, namely the number $K$ of neighbors in the final nearest neighbor rule. This parameter is common to all nearest neighbor classification techniques.

## 4  Experimental Results

In the following we compare several classification methods using both simulated and real data. We compare the following classification approaches: (1) LFM-SVM algorithm described in Figure 1. $SVM^{light}$ [12] with radial basis kernels is used to build the SVM classifier; (2) RBF-SVM classifier with radial basis kernels. We used $SVM^{light}$ [12], and set the value of $\gamma$ in $K(\mathbf{x}_i, \mathbf{x}) = e^{-\gamma\|\mathbf{x}_i - \mathbf{x}\|^2}$ equal to the optimal one determined via cross-validation. Also the value of $C$ for the soft-margin classifier is optimized via cross-validation. The output of this classifier is the input of LFM-SVM; (3) ADAMENN-adaptive metric nearest neighbor technique [7]. It uses the *Chi-squared* distance in order to estimate to which extent each dimension can be relied on to predict class posterior probabilities; (4) Machete [9]. It is a recursive partitioning procedure, in which the input variable used for splitting at each step is the one that maximizes the estimated local relevance. Such relevance is measured in terms of the improvement in squared prediction error each feature is capable to provide; (5) Scythe [9]. It is a generalization of the machete algorithm, in which the input variables influence each split in proportion to their estimated local relevance; (6) DANN-discriminant adaptive nearest neighbor classification [10]. It is an adaptive nearest neighbor classification method based on linear discriminant analysis. It computes a distance metric as a product of properly weighted within and between sum of squares matrices; (7) Simple K-NN method using the Euclidean distance measure; (8) C4.5 decision tree method [15].

In all the experiments, the features are first normalized over the training data to have zero mean and unit variance, and the test data features are normalized using the corresponding training mean and variance. Procedural parameters (including

$K$) for each method were determined empirically through cross-validation.

## 4.1 Experiments on Simulated Data

For all simulated data, 10 independent training samples of size 200 were generated. For each of these, an additional independent test sample consisting of 200 observations was generated. These test data were classified by each competing method using the respective training data set. Error rates computed over all 2,000 such classifications are reported in Table 1.

**The Problems.** (1) **Multi-Gaussians.** The data set consists of $n = 2$ input features, $l = 200$ training data, and $J = 2$ classes. Each class contains two spherical bivariate normal subclasses, having standard deviation 1. The mean vectors for one class are $(-3/4, -3)$ and $(3/4, 3)$; whereas for the other class are $(3, -3)$ and $(-3, 3)$. For each class, data are evenly drawn from each of the two normal subclasses. The first column of Table 1 shows the results for this problem. The standard deviations are: 0.17, 0.01, 0.01, 0.01, 0.01 0.01, 0.01 and 1.50, respectively. (2) **Noisy-Gaussians.** The data for this problem are generated as in the previous example, but augmented with four predictors having independent standard Gaussian distributions. They serve as noise. Results are shown in the second column of Table 1. The standard deviations are: 0.18, 0.01, 0.02, 0.01, 0.01, 0.01, 0.01 and 1.60, respectively.

**Results.** Table 1 shows that all methods have similar performances for the Multi-Gaussians problem, with C4.5 being the worst performer. When the noisy predictors are added to the problem (NoisyGaussians), we observe different levels of deterioration in performance among the eight methods. LFM-SVM shows the most robust behavior in presence of noise. K-NN is instead the worst performer. In Figure 2 we plot the performances of LFM-SVM and RBF-SVM as a function of an increasing number of noisy features (for the same MultiGaussians problem). The standard deviations for RBF-SVM (in order of increasing number of noisy features) are: 0.01, 0.01, 0.03, 0.03, 0.03 and 0.03. The standard deviations for LFM-SVM are: 0.17, 0.18, 0.2, 0.3, 0.3 and 0.3. The LFM-SVM technique shows a considerable improvement over RBF-SVM as the amount of noise increases.

Table 1: Average classification error rates for simulated and real data.

|  | MultiGauss | NoisyGauss | Iris | Sonar | Liver | Vote | Breast | OQ | Pima |
|---|---|---|---|---|---|---|---|---|---|
| LFM-SVM | **3.3** | **3.4** | 4.0 | 11.0 | 28.1 | **2.6** | 3.0 | 3.5 | **19.3** |
| RBF-SVM | **3.3** | 4.1 | 4.0 | 12.0 | **26.1** | 3.0 | 3.1 | 3.4 | 21.3 |
| ADAMENN | 3.4 | 4.1 | **3.0** | 9.1 | 30.7 | 3.0 | 3.2 | **3.1** | 20.4 |
| Machete | 3.4 | 4.3 | 5.0 | 21.2 | 27.5 | 3.4 | 3.5 | 7.4 | 20.4 |
| Scythe | 3.4 | 4.8 | 4.0 | 16.3 | 27.5 | 3.4 | 2.7 | 5.0 | 20.0 |
| DANN | 3.7 | 4.7 | 6.0 | **7.7** | 30.1 | 3.0 | **2.2** | 4.0 | 22.2 |
| K-NN | **3.3** | 7.0 | 6.0 | 12.5 | 32.5 | 7.8 | 2.7 | 5.4 | 24.2 |
| C4.5 | 5.0 | 5.1 | 8.0 | 23.1 | 38.3 | 3.4 | 4.1 | 9.2 | 23.8 |

## 4.2 Experiments on Real Data

In our experiments we used seven different real data sets. They are all taken from UCI Machine Learning Repository at http://www.cs.uci.edu/~mlearn/MLRepository.html. For a description of the data sets see [6]. For the Iris, Sonar, Liver and Vote data we perform leave-one-out cross-validation to measure performance, since the number of available data is limited for these data sets. For the

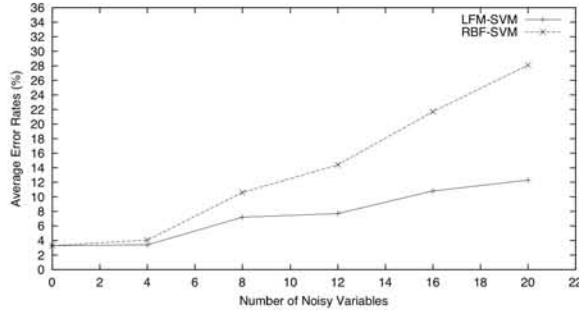

Figure 2: Average Error Rates of LFM-SVM and RBF-SVM as a function of an increasing number of noisy predictors.

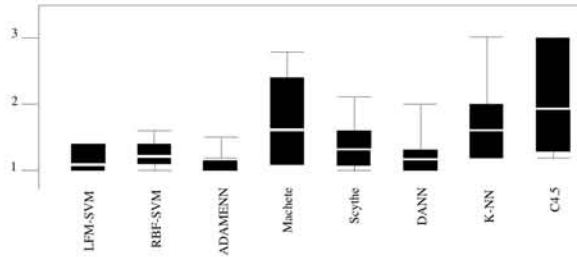

Figure 3: Performance distributions for real data.

Breast, OQ-letter and Pima data we randomly generated five independent training sets of size 200. For each of these, an additional independent test sample consisting of 200 observations was generated. Table 1 (columns 3-9) shows the cross-validated error rates for the eight methods under consideration on the seven real data. The standard deviation values are as follows. Breast data: 0.2, 0.2, 0.2, 0.2, 0.2, 0.9, 0.9 and 0.9, respectively. OQ data: 0.2, 0.2, 0.2, 0.3, 0.2, 1.1, 1.5 and 2.1, respectively. Pima data: 0.4, 0.4, 0.4, 0.4, 0.4, 2.4, 2.1 and 0.7, respectively.

**Results**. Table 1 shows that LFM-SVM achieves the best performance in 2/7 of the real data sets; in one case it shows the second best performance, and in the remaining four its error rate is still quite close to the best one. Following Friedman [9], we capture robustness by computing the ratio $b_m$ of the error rate $e_m$ of method $m$ and the smallest error rate over all methods being compared in a particular example: $b_m = e_m / \min_{1 \le k \le 8} e_k$.

Figure 3 plots the distribution of $b_m$ for each method over the seven real data sets. The dark area represents the lower and upper quartiles of the distribution that are separated by the median. The outer vertical lines show the entire range of values for the distribution. The spread of the error distribution for LFM-SVM is narrow and close to one. The results clearly demonstrate that LFM-SVM (and ADAMENN) obtained the most robust performance over the data sets.

The poor performance of the machete and C4.5 methods might be due to the greedy strategy they employ. Such recursive peeling strategy removes at each step a subset of data points permanently from further consideration. As a result, changes in an early split, due to any variability in parameter estimates, can have a significant impact on later splits, thereby producing different terminal regions. This makes

predictions highly sensitive to the sampling fluctuations associated with the random nature of the process that produces the traning data, thus leading to high variance predictions. The scythe algorithm, by relaxing the winner-take-all splitting strategy of the machete algorithm, mitigates the greedy nature of the approach, and thereby achieves better performance.

In [10], the authors show that the metric employed by the DANN algorithm approximates the weighted Chi-squared distance, given that class densities are Gaussian and have the same covariance matrix. As a consequence, we may expect a degradation in performance when the data do not follow Gaussian distributions and are corrupted by noise, which is likely the case in real scenarios like the ones tested here.

We observe that the sparse solution given by SVMs provides LFM-SVM with principled guidelines to efficiently set the input parameters. This is an important advantage over ADAMENN, which has six tunable input parameters. Furthermore, LFM-SVM speeds up the classification process since it applies the nearest neighbor rule only once, whereas ADAMENN applies it at each point within a region centered at the query. We also observe that the construction of the SVM for LFM-SVM is carried out off-line only once, and there exist algorithmic and computational results which make SVM training practical also for large-scale problems [12].

The LFM-SVM offers performance improvements over the RBF-SVM algorithm alone, for both the (noisy) simulated and real data sets. The reason for such performance gain may rely on the effect of our local weighting scheme on the separability of classes, and therefore on the margin, as shown in [6]. Assigning large weights to input features close to the gradient direction, locally in neighborhoods of support vectors, corresponds to increase the spatial resolution along those orientations, and therefore to improve the separability of classes. As a consequence, better classification results can be achieved as demonstrated in our experiments.

## 5 Related Work

In [1], Amari and Wu improve support vector machine classifiers by modifying kernel functions. A primary kernel is first used to obtain support vectors. The kernel is then modified in a data dependent way by using the support vectors: the factor that drives the transformation has larger values at positions close to support vectors. The modified kernel enlarges the spatial resolution around the boundary so that the separability of classes is increased.

The resulting transformation depends on the distance of data points from the support vectors, and it is therefore a local transformation, but is independent of the boundary's orientation in input space. Likewise, our transformation metric depends, through the factor $A$, on the distance of the query point from the support vectors. Moreover, since we weight features, our metric is directional, and depends on the orientation of local boundaries in input space. This dependence is driven by our measure of feature relevance, which has the effect of increasing the spatial resolution along discriminant directions around the boundary.

## 6 Conclusions

We have described a locally adaptive metric classification method and demonstrated its efficacy through experimental results. The proposed technique offers performance improvements over the SVM alone, and has the potential of scaling up to

large data sets. It speeds up, in fact, the classification process by computing off-line the information relevant to define local weights, and by applying the nearest neighbor rule only once.

**Acknowledgments**

This research has been supported by the National Science Foundation under grants NSF CAREER Award 9984729 and NSF IIS-9907477, by the US Department of Defense, and a research award from AT&T.

# References

[1] S. Amari and S. Wu, "Improving support vector machine classifiers by modifying kernel functions", *Neural Networks*, 12, pp. 783-789, 1999.

[2] R.E. Bellman, *Adaptive Control Processes*. Princeton Univ. Press, 1961.

[3] M. Brown, W. Grundy, D. Lin, N. Cristianini, C. Sugnet, T. Furey, M. Ares, and D. Haussler, "Knowledge-based analysis of microarray gene expressions data using support vector machines", Tech. Report, University of California in Santa Cruz, 1999.

[4] W.S. Cleveland and S.J. Devlin, "Locally Weighted Regression: An Approach to Regression Analysis by Local Fitting", *J. Amer. Statist. Assoc.* **83**, 596-610, 1988

[5] T.M. Cover and P.E. Hart, "Nearest Neighbor Pattern Classification", *IEEE Trans. on Information Theory*, pp. 21-27, 1967.

[6] C. Domeniconi and D. Gunopulos, "Adaptive Nearest Neighbor Classification using Support Vector Machines", Tech. Report UCR-CSE-01-04, Dept. of Computer Science, University of California, Riverside, June 2001.

[7] C. Domeniconi, J. Peng, and D. Gunopulos, "An Adaptive Metric Machine for Pattern Classification", *Advances in Neural Information Processing Systems*, 2000.

[8] R.O. Duda and P.E. Hart, *Pattern Classification and Scene Analysis*. John Wiley & Sons, Inc., 1973.

[9] J.H. Friedman "Flexible Metric Nearest Neighbor Classification", Tech. Report, Dept. of Statistics, Stanford University, 1994.

[10] T. Hastie and R. Tibshirani, "Discriminant Adaptive Nearest Neighbor Classification", *IEEE Trans. on Pattern Analysis and Machine Intelligence*, Vol. 18, No. 6, pp. 607-615, 1996.

[11] T. Joachims, "Text categorization with support vector machines", *Proc. of European Conference on Machine Learning*, 1998.

[12] T. Joachims, "Making large-scale SVM learning practical" *Advances in Kernel Methods - Support Vector Learning*, B. Schölkopf and C. Burger and A. Smola (ed.), MIT-Press, 1999. http://www-ai.cs.uni-dortmund.de/thorsten/svm_light.html

[13] D.G. Lowe, "Similarity Metric Learning for a Variable-Kernel Classifier", *Neural Computation* **7**(1):72-85, 1995.

[14] E. Osuna, R. Freund, and F. Girosi, "Training support vector machines: An application to face detection", *Proc. of Computer Vision and Pattern Recognition*, 1997.

[15] J.R. Quinlan, *C4.5: Programs for Machine Learning*. Morgan-Kaufmann Publishers, Inc., 1993.

[16] C.J. Stone, Nonparametric regression and its applications (with discussion). *Ann. Statist.* **5**, 595, 1977.
